# Boosting with Spatial Regularization

**Zhen James Xiang**[1]    **Yongxin Taylor Xi**[1]    **Uri Hasson**[2]    **Peter J. Ramadge**[1]

1: Department of Electrical Engineering, Princeton University, Princeton NJ, USA
2: Department of Psychology, and Neuroscience Institute, Princeton University, Princeton NJ, USA
{zxiang, yxi, hasson, ramadge} @ princeton.edu

## Abstract

By adding a spatial regularization kernel to a standard loss function formulation of the boosting problem, we develop a framework for spatially informed boosting. From this regularized loss framework we derive an efficient boosting algorithm that uses additional weights/priors on the base classifiers. We prove that the proposed algorithm exhibits a "grouping effect", which encourages the selection of all spatially local, discriminative base classifiers. The algorithm's primary advantage is in applications where the trained classifier is used to identify the spatial pattern of discriminative information, e.g. the voxel selection problem in fMRI. We demonstrate the algorithm's performance on various data sets.

## 1 Introduction

When applying off-the-shelf machine learning algorithms to data with spatial dimensions (images, geo-spatial data, fMRI, etc) a central question arises: how to incorporate prior information on the spatial characteristics of the data? For example, if we feed a boosting or SVM algorithm with individual image voxels as features, the voxel spatial information is ignored. Indeed, if we randomly shuffled the voxels, the algorithm would not notice any difference. Yet in many cases the spatial arrangement of the voxels together with prior information about expected spatial characteristics of the data may be very helpful. We are particularly interested in the situation when the trained classifier is used to *identify* relevant spatial regions. To make this more concrete, consider the problem of training a classifier to distinguish two different brain states based on fMRI responses. Successful classification suggests that the voxels used are important in discriminating between the two classes. Hence we could use a successful classifier to learn a set of discriminative voxels. We expect that these voxels will be spatially compact and clustered. How can this prior knowledge be incorporated into the training of the classifier? In summary, our primary objective is improving the ability of the trained classifier to usefully identify the spatial pattern of discriminative information. However, incorporating spatial information into boosting may also improve classification accuracy.

Our key contribution is the development of a framework for spatially regularized boosting. We do this by adding a spatial regularization kernel to the standard loss minimization formulation of boosting. We then design an associated boosting algorithm by using coordinate descent on the regularized loss. We show that the algorithm minimizes the regularized loss function and has a natural interpretation of boosting with additional adaptive priors/weights on *both* spatial locations and training examples. We also show that it exhibits a natural grouping effect on nearby spatial locations with similar discriminative power.

We believe our contributions are fundamental and relevant to a variety of applications where base classifiers are attributed with a known auxiliary variable and prior information is known about this auxiliary variable. However, since our study is motivated by the particular problem of voxel selection in fMRI analysis, we briefly review the state of the art in this domain so as to put our contribution into a concrete context.

Briefly, the fMRI voxel selection problem is to use the fMRI signal to identify a subset of voxels that are key in discriminating between two stimuli. One expects such voxels to be spatially compact and clustered. Traditionally this is done by thresholding a statistical univariate test score on each voxel [1]. Spatial smoothing prior to this analysis is commonly employed to integrate activity from neighboring voxels. An extreme case is hypothesis testings on clusters of voxels rather than on voxels themselves [2]. The problem with these methods is that they greatly sacrifice the spatial resolution of the results and averaging could hide fine patterns in data. An alternative is to spatially average the univariate test scores, e.g. thresholding in some transformed domain (e.g. wavelet domain) [3, 4]. However, this also compromises the spatial accuracy of the result because one selects discriminating wavelet components, not voxels. A more promising spatially aware approach selects voxels with tree-based spatial regularization of a univariate statistic [5, 6]. This can achieve both spatial precision and smoothness but uses a complex regularization method. Our proposed method also selects single voxels with the help of spatial regularization but operates in a multivariate classifier framework using a simpler form of regularization.

Recent research has suggested that *multivariate* analysis has potential advantages over univariate tests [7, 8], e.g. it brings in machine learning algorithms (such as boosting, SVM, etc.) and therefore might capture more intricate activation patterns involving multiple voxels. To ensure spatial clustering of selected voxels, one can run a searchlight (a spherical mask) [9] to pre-select clustered informative features. In each searchlight location, a multivariate analysis is performed to see whether the masked area contains informative data. One can then train a classifier on the pre-selected voxels. A variant of this two-stage framework is to train classifiers on a few predefined masks, and then aggregate these classifiers by boosting [10, 11]. This is faster but assumes detailed prior knowledge to select the predefined masks. Unlike two-stage approaches, [12] directly uses AdaBoost to train classifiers with "rich features" (features involving the values of several adjacent voxels) to capture spatial structure in the data. Although exhibiting superior performance, this method selects "rich features" rather than individual discriminating voxels. Moreover, there is no control on the spatial smoothness of the results. Our method is similar to [12] in that we combine the feature selection and classification into one boosting process. But our algorithm operates on single voxels and uses simple spatial regularization to incorporate spatial information.

The remainder of the paper is organized as follows. After introducing notation in §2, we formulate our spatial regularization approach in §3 and derive an associated spatially regularized boosting algorithm in §4. We prove an interesting property of the algorithm in §5 that guarantees the simultaneous selection of equivalent locations that are spatially close. In §6, we test the algorithm on face gender detection, OCR image classification, and fMRI experiments.

## 2 Boosting Preliminaries

In a supervised learning setting, we are given $m$ training instances $\mathcal{X} = \{x_i \in \mathbb{R}^n, i = 1, \ldots, m\}$ and corresponding binary labels $\mathcal{Y} = \{y_i = \pm 1, i = 1, \ldots, m\}$. Using the training instances $\mathcal{X}$, we select a pool of base classifiers $\mathcal{H} = \{h_j \colon \mathbb{R}^n \to \{-1, +1\}, j = 1, \ldots, p\}$. Our objective is to train a composite binary classifier of the form $h_{\boldsymbol{\alpha}}(x_i) = \operatorname{sgn}(\sum_{j=1}^{p} \alpha_j h_j(x_i))$. We can further assume that $h_j \in \mathcal{H} \Rightarrow -h_j \in \mathcal{H}$, thus all values in $\boldsymbol{\alpha}$ can be assumed to be nonnegative. Boosting is a technique for constructing from $\mathcal{X}, \mathcal{Y}$ and $\mathcal{H}$ the weight $\boldsymbol{\alpha}$ of a composite classifier to best predict the labels. This can be done by seeking $\boldsymbol{\alpha}$ to minimize a loss function of the form:

$$\mathcal{L}(\mathcal{X}, \mathcal{Y}, \boldsymbol{\alpha}) = \sum_{i=1}^{m} l(y_i, h_{\boldsymbol{\alpha}}(x_i)). \tag{1}$$

Various boosting algorithms can be derived as iterative greedy coordinate descent procedures to minimize (1) [13]. In particular, AdaBoost [14] is of this form with $l(y_i, h_{\boldsymbol{\alpha}}(x_i)) = e^{-y_i h_{\boldsymbol{\alpha}}(x_i)}$.

The result of a conventional boosting algorithm is determined by the $m \times p$ matrix $M = [y_i h_j(x_i)]$ [15]. Under a component permutation $\hat{x}_i = P x_i$, the base classifiers become $\hat{h}_j = h_j \cdot P^{-1}$; so $\hat{M} = [y_i \hat{h}_j(\hat{x}_i)] = [y_i h_j(x_i)] = M$. Hence training on $\{P x_i, y_i\}$ or $\{x_i, y_i\}$ yields the same $\boldsymbol{\alpha}$, i.e., the arrangement of the components can be arbitrary as long as it is consistent.

The weights $\boldsymbol{\alpha}$ of a composite classifier not only indicate how to construct the classifier, but also the relative reliance of the classifier on each of the $n$ instance components. To see this, assume each

$h_j$ depends on only a single component of $x \in \mathbb{R}^n$, i.e., for some standard basis vector $e_k$, and function $g_j \colon \mathbb{R} \to \{-1, +1\}$, $h_j(x) = g_j(e_k^T x)$ (the base classifiers are decision stumps). To make the association between base classifiers and components explicit, let $s$ be the function $s(j) = k$ if $h_j(x) = g_j(e_k^T x)$ and $Q = [q_{kj}]$ be the $n \times p$ matrix with $q_{kj} = \mathbb{1}_{[s(j)=k]}$. Then the vector $\boldsymbol{\beta} = Q\boldsymbol{\alpha}$ indicates the relative importance the classifier assigns to each instance component. Although we used decision stumps above for simplicity, more complex base classifiers such as decision trees could be used with proper modification of mapping from $\boldsymbol{\alpha}$ to $\boldsymbol{\beta}$. We call $\boldsymbol{\beta}$ the *component importance map*. Suppose the instance components reflect spatial structure in the data, e.g. the components are samples along an interval or pixels in an image. Then the component importance map is indicating the spatial distribution of weights that the classifier employs. Presumably a good classifier distributes the weights in accordance with the discriminative power of the components; in which case, the map is indicating how discriminative information is spatially distributed. It is in this aspect of the classifier that we are particularly interested. Now as shown above, conventional boosting ignores spatial information. Our objective, pursued in the next sections, is to incorporated prior information on spatial structure, e.g. a prior on the component importance map, into the boosting problem.

## 3 Adding Spatial Regularization

To incorporate spatial information we add spatial regularization of the form $\boldsymbol{\beta}^T K \boldsymbol{\beta}$ to the loss (1) where the kernel $K \in \mathbb{R}^{n \times n}_{++}$ is positive definite. For concreteness, we employ the exponential loss $l(y_i, h_{\boldsymbol{\alpha}}(x_i)) = e^{-y_i h_{\boldsymbol{\alpha}}(x_i)}$. Thus the regularized loss is:

$$\mathcal{L}_{reg}^{exp}(\mathcal{X}, \mathcal{Y}, \boldsymbol{\alpha}) = \sum_{i=1}^{m} \exp(-y_i \sum_{j=1}^{p} \alpha_j h_j(x_i)) + \lambda \boldsymbol{\beta}^T K \boldsymbol{\beta} \tag{2}$$

$$= \sum_{i=1}^{m} \exp(-y_i \sum_{j=1}^{p} \alpha_j h_j(x_i)) + \lambda \boldsymbol{\alpha}^T Q^T K Q \boldsymbol{\alpha}. \tag{3}$$

The term $\boldsymbol{\beta}^T K \boldsymbol{\beta}$ imposes a spatial smoothness constraint on $\boldsymbol{\beta}$. To see this, consider the eigen-decomposition $K = U \Sigma U^T$, where the columns $\{u_j\}$ of $U$ are the orthonormal eigenvectors, $\sigma_j$ is the eigenvalue of $u_j$ and $\Sigma = \mathrm{diag}(\sigma_1, \sigma_2, \ldots, \sigma_n)$. Then the regularizing term can be rewritten as $\lambda \|\Sigma^{\frac{1}{2}} U^T \boldsymbol{\beta}\|_2^2$ where $U^T \boldsymbol{\beta}$ is the "spectrum" of $\boldsymbol{\beta}$ under the orthogonal transformation $U^T$. Rather than standard Tikhonov regularization with $\|\boldsymbol{\beta}\|_2^2 = \|U^T \boldsymbol{\beta}\|_2^2$, we penalize the variation in direction $u_j$ proportional to the eigenvalue $\sigma_j$. By doing so we are encouraging $\boldsymbol{\beta}$ to be close to the eigenvectors $u_j$ with small eigenvalues. This encodes our prior spatial knowledge.

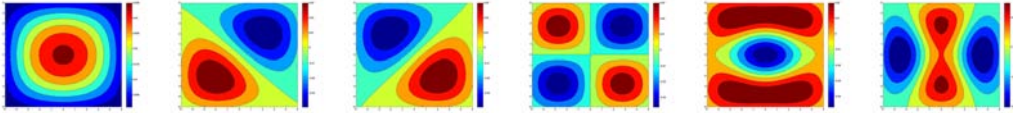

*Figure 1:* Each graph is the eigenimage of size $d \times d$ corresponding to an eigenvector of $K = \mu I - G$.

As an example, consider the kernel $K = \mu I - G$, where $G$ is a Gaussian kernel matrix:

$$G_{ij} = e^{-\frac{1}{2} \|v_i - v_j\|_2^2 / r^2}, \tag{4}$$

with $v_j$ the spatial location of component $j$, $\|v_i - v_j\|_2$ the Euclidean distance (other distances can also be used) between components $i$ and $j$, and $r$ the radius parameter of the Gaussian kernel. For the 2D case, $i = (i_1, i_2)$ ranges over $(1, 1), (1, 2), \ldots, (d, d)$. $j = (j_1, j_2)$ ranges over the same coordinates. So $G$ is a size $d^2 \times d^2$ matrix. We plot the 6 eigenimages of $K$ with smallest eigenvalues in Figure 1. The regularization imposes a spatial smoothness constraint by encouraging $\boldsymbol{\beta}$ to give more weight to the eigenimages with smaller eigenvalues, e.g. the patterns shown in Figure 1.

## 4 A Spatially Regularized Boosting Algorithm

We now derive a spatially regularized boosting algorithm (abbreviated as SRB) using coordinate descent on (3). In particular, in each iteration we choose a coordinate of $\boldsymbol{\alpha}$ with the largest negative

gradient and increase the weight of that coordinate by step size $\varepsilon$. This results in an algorithm similar to AdaBoost, but with additional consideration of spatial location.

To begin, we take the partial derivative of (3) w.r.t. $\alpha_{j'}$:

$$-\frac{\partial}{\partial \alpha_{j'}} \mathcal{L}_{reg}^{exp}(\mathcal{X}, \mathcal{Y}, \boldsymbol{\alpha}) = \sum_{i=1}^{m} y_i h_{j'}(x_i) \exp(-y_i \sum_{j=1}^{p} \alpha_j h_j(x_i)) - 2e_{j'}^T \lambda Q^T K Q \boldsymbol{\alpha}.$$

Here $e_{j'}$ is the $j'$-th standard basis vector, so $e_{j'}^T \lambda Q^T K Q \boldsymbol{\alpha}$ is the $j'$-th element of $\lambda Q^T K Q \boldsymbol{\alpha}$. By the definition of $Q$, $(e_{j'}^T Q^T) \lambda K Q \boldsymbol{\alpha}$ is the $s(j')$-th element of $\lambda K Q \boldsymbol{\alpha}$. Therefore if we define $\boldsymbol{\gamma}$ to be $\boldsymbol{\gamma} = -2\lambda K \boldsymbol{\beta}$, and $w_i = \exp(-y_i \sum_{j=1}^{p} \alpha_j h_j(x_i))$ $(1 \le i \le m)$ to be the unnormalized weight on training instance $x_i$, then the partial derivative in (4) can be written as:

$$-\frac{\partial}{\partial \alpha_{j'}} \mathcal{L}_{reg}^{exp}(\mathcal{X}, \mathcal{Y}, \boldsymbol{\alpha}) = \sum_{i=1}^{m} y_i h_{j'}(x_i) w_i + \gamma_{s(j')}$$

The term $\sum_{i=1}^{m} y_i h_{j'}(x_i) w_i$ is the weighted performance of base classifier $h_{j'}$ on the training examples. Normally, we choose $h_{j'}$ to maximize this term. This corresponds to choosing the best base classifier under the current weight distribution. However, here we have an additional term: the performance of base classifier $h_{j'}$ is enhanced by a weight $\gamma_{s(j')}$ on its corresponding component $s(j')$. We call $\boldsymbol{\gamma}$ the *spatial compensation weight*. To proceed, we choose a base classifier $h_{j'}$ to maximize the sum of these two terms and then increase the weight of that base classifier by a step size $\varepsilon$. This gives Algorithm 1 shown in Figure 2. The key differences from AdaBoost are: (a) the new algorithm maintains a new set of "spatial compensation weights" $\boldsymbol{\gamma}$; (b) the weights on training examples $w_i$ are not normalized at the end of each iteration.

---

**Algorithm 1** The SRB algorithm

1: $w_i \leftarrow 1, 1 \le i \le m$
2: $\boldsymbol{\alpha} \leftarrow \mathbf{0}$
3: **for** $t = 1$ to $T$ **do**
4:     $\boldsymbol{\beta} \leftarrow Q\boldsymbol{\alpha}$
5:     $\boldsymbol{\gamma} \leftarrow -2\lambda K \boldsymbol{\beta}$
6:     find the "best" base classifier in the following sense:
    $j' \leftarrow \arg\max_j \{\Omega(h_j, \boldsymbol{w}) + \gamma_{s(j)}\}$
7:     choose a step size $\varepsilon$, $\alpha_{j'} \leftarrow \alpha_{j'} + \varepsilon$
8:     adjust weights:
$$w_i \leftarrow \begin{cases} w_i e^{\varepsilon} & \text{if } y_i h_{j'}(x_i) = -1 \\ w_i e^{-\varepsilon} & \text{if } y_i h_{j'}(x_i) = 1 \end{cases}$$
    for $1 \le i \le m$
9: **end for**
10: Output result: $h_{\boldsymbol{\alpha}}(x) = \sum_{j=1}^{p} \alpha_j h_j(x)$

---

*In both algorithms, $\Omega(h_j, \boldsymbol{w})$ is defined to be:*

$$\Omega(h_j, \boldsymbol{w}) = \sum_{i=1}^{m} y_i h_j(x_i) w_i,$$

*which is a performance measure of classifier $h_j$ under weight distribution $\boldsymbol{w}$ on training examples.*

---

**Algorithm 2** SRB algorithm with backward steps

1: $w_i \leftarrow 1, 1 \le i \le m$
2: $\boldsymbol{\alpha} \leftarrow \mathbf{0}$
3: **for** $t = 1$ to $T$ **do**
4:     $\boldsymbol{\beta} \leftarrow Q\boldsymbol{\alpha}$
5:     $\boldsymbol{\gamma} \leftarrow -2\lambda K \boldsymbol{\beta}$
6:     find the "best" base classifier in the following sense:
    $j' \leftarrow \arg\max_j \{\Omega(h_j, \boldsymbol{w}) + \gamma_{s(j)}\}$
7:     choose a step size $\varepsilon_1$, $\alpha_{j'} \leftarrow \alpha_{j'} + \varepsilon_1$
8:     adjust weights:
$$w_i \leftarrow \begin{cases} w_i e^{\varepsilon_1} & \text{if } y_i h_{j'}(x_i) = -1 \\ w_i e^{-\varepsilon_1} & \text{if } y_i h_{j'}(x_i) = 1 \end{cases}$$
9:     find the "worst" active classifier in the following sense:
    $j'' \leftarrow \arg\min_{j:\alpha_j>0} \{\Omega(h_j, \boldsymbol{w}) + \gamma_{s(j)}\}$
10:    $\alpha_{j''} \leftarrow \alpha_{j''} - \frac{\varepsilon_2}{2}$
11:    adjust weights again:
$$w_i \leftarrow \begin{cases} w_i e^{-\varepsilon_2/2} & \text{if } y_i h_{j''}(x_i) = -1 \\ w_i e^{\varepsilon_2/2} & \text{if } y_i h_{j''}(x_i) = 1 \end{cases}$$
   for $1 \le i \le m$
12: **end for**
13: Output result: $h_{\boldsymbol{\alpha}}(x) = \sum_{j=1}^{p} \alpha_j h_j(x)$

---

*Figure 2:* The SRB (spatially regularized boosting algorithms).

To elucidate the effect of the compensation weights, consider the kernel $K = \mu I - G$, with $G$ defined in (4). In this case, $\boldsymbol{\gamma} = 2\lambda(\bar{\boldsymbol{\beta}} - \mu\boldsymbol{\beta})$ where $\bar{\boldsymbol{\beta}} = G\boldsymbol{\beta}$ is the Gaussian smoothing of $\boldsymbol{\beta}$. Therefore,

a component receives a high compensation weight $\gamma_k = 2\lambda(\bar{\beta}_k - \mu\beta_k)$ if some neighboring spatial locations have already been selected (i.e., made "active") by the composite classifier. On the other hand, the weight of a component is reduced (proportional to the magnitude of parameter $\mu$) if it is already "active", i.e., $\beta_k > 0$. So the algorithm encourages the selection of base classifiers associated with "inactive" locations that are close to "active" locations.

We can enhance the algorithm by including a backward step each iteration: $\alpha_{j''} \leftarrow \alpha_{j''} - \varepsilon'$, where

$$j'' = \underset{1 \leq j \leq p, \alpha_j > 0}{\arg\min} \left\{ \sum_{i=1}^m y_i h_j(x_i) w_i + \gamma_{s(j)} \right\}. \tag{5}$$

This helps remove prematurely selected base classifiers [16, 17]. This is Algorithm 2 in Figure 2.

Spatial regularization brings no significant computational overhead: Compared to AdaBoost, SRB has additional steps 4,5, which can be computed in time $O(n)$ every iteration. Adaptive weight $\gamma$ incurs no additional complexity for step 6 in our current implementation.

We now briefly discuss the choice of step size $\varepsilon$ in Algorithm 1 ($\varepsilon_1$ and $\varepsilon_2$ in Algorithm 2 can be chosen similarly). $\varepsilon$ could be a fixed (small) step size at each iteration. This is not greedy but may necessitate a large number of iterations. Alternatively, one can be greedy and select $\varepsilon$ to minimize the value of the loss function (3) after the change $\alpha_{j'} \leftarrow \alpha_{j'} + \varepsilon$:

$$W_- e^\varepsilon + W_+ e^{-\varepsilon} + \lambda(\boldsymbol{\beta} + \varepsilon e_{k'})^T K(\boldsymbol{\beta} + \varepsilon e_{k'}), \tag{6}$$

where $W_- = \sum_{i:y_i h_{j'}(x_i)=-1} \exp(-y_i h_{\boldsymbol{\alpha}}(x_i))$, $W_+ = \sum_{i:y_i h_{j'}(x_i)=1} \exp(-y_i h_{\boldsymbol{\alpha}}(x_i))$ and $k' = s(j')$. Setting the derivative of (6) to 0 yields:

$$W_- e^\varepsilon - W_+ e^{-\varepsilon} - \gamma_{k'} + 2\lambda\varepsilon K_{k'k'} = 0. \tag{7}$$

Using $e^{\pm\varepsilon} \approx 1 \pm \varepsilon$ gives the solution $\hat{\varepsilon} = \frac{W_+ - W_- + \gamma_{k'}}{W_+ + W_- + 2\lambda K_{k'k'}}$, which can be used as a step size. However, for the following slightly more conservative step size we can prove algorithm convergence:

$$\tilde{\varepsilon} = \min \left\{ 3\frac{(W_+ - W_-)}{W_+ + 1.36W_-}, \frac{W_+ - W_- + \gamma_{k'}}{W_+ + W_- + 2\lambda K_{k'k'}}, 1 \right\}. \tag{8}$$

**Theorem 1.** *The step size* (8) *ensures convergence of Algorithm 1.*

*Proof.* (6) is convex, so its minimum point $\varepsilon^*$ is the unique solution of (7): $f_1(\varepsilon^*) + f_2(\varepsilon^*) = 0$ where $f_1(\varepsilon) = W_- e^\varepsilon - W_+ e^{-\varepsilon}$ and $f_2(\varepsilon) = 2\lambda K_{k'k'}\varepsilon - \gamma_{k'}$. We have the inequality chain:

$$f_1(\tilde{\varepsilon}) + f_2(\tilde{\varepsilon}) \leq g_1(\tilde{\varepsilon}) + f_2(\tilde{\varepsilon}) \leq g_1(\hat{\varepsilon}) + f_2(\hat{\varepsilon}) = 0 = f_1(\varepsilon^*) + f_2(\varepsilon^*), \tag{9}$$

where $g_1(\varepsilon) = W_-(1+\varepsilon) - W_+(1-\varepsilon)$. So $\tilde{\varepsilon}$ is on the descending slope of (6), which is a sufficient condition for $\tilde{\varepsilon}$ to reduce the objective (6). Since the objective (3) is nonnegative and each iteration of the algorithm reduces (3), the algorithm converges. The second inequality in (9) uses monoticity while the first inequality in (9) uses the following lemma proved in the supplementary material:
**Lemma:** If $0 < \varepsilon \leq \min\{3\frac{(W_+ - W_-)}{W_+ + 1.36W_-}, 1\}$, then $f_1(\varepsilon) - g_1(\varepsilon) \leq 0$. $\qquad\square$

## 5  The Grouping Effect: Asymptotic Analysis

Recall our objective of using the component importance map of the trained classifier to ascertain the spatial distribution of informative components in the data. Ideally, we would like $\beta$ to faithfully represent this information. In general, however, a boosting algorithm will select a *sufficient* but *incomplete* collection of base classifiers (and hence components) to accomplish the classification. For example, after selecting one base classifier $h_j$, AdaBoost will adjust the weights of training examples to make the weighted training error of $h_j$ exactly $\frac{1}{2}$ (totally uninformative), thus preventing the selection of any classifiers similar to $h_j$ in the next iteration. In fact, for AdaBoost we can prove that in the optimal solution $\boldsymbol{\alpha}^*$, we can transfer coefficient weights between any two equivalent base classifiers without impacting optimality. So minimizing the loss function (1) does not require any particular distribution among the $\boldsymbol{\beta}$ coefficients of identical components. This is the content of the following proposition.

**Proposition 1.** *Assume $h_{j_1}$ and $h_{j_2}$, $j_1 < j_2$, are base classifiers with $s(j_1) \neq s(j_2)$, and $h_{j_1}(x_i) = h_{j_2}(x_i)$ for all $x_i \in \mathcal{X}$. If $\boldsymbol{\alpha}^*$ minimizes the loss function* (1)*, then for any $\eta$ in $[0, \min\{\alpha^*_{j_1}, \alpha^*_{j_2}\}]$, $\boldsymbol{\alpha}^\dagger$ also minimizes loss function* (1) *where $\boldsymbol{\alpha}^\dagger = \boldsymbol{\alpha}^* - \eta e_{j_1} + \eta e_{j_2}$ where $e_j$ denotes the $j$-th standard basis vector in $\mathbb{R}^p$.*

*Proof.* $h_{j_1}(x_i) = h_{j_2}(x_i)$ implies that $h_{\boldsymbol{\alpha}^*}(x_i) = h_{\boldsymbol{\alpha}^\dagger}(x_i)$ for all $x_i \in \mathcal{X}$. $\qquad\square$

What is desirable is a "grouping effect", in which components with similar behavior under $\mathcal{H}$ receive similar $\beta$ weights. We will prove that asymptotically, SRB exhibits a "grouping effect". In particular, for kernel $K = \mu I - G$, $G$ defined in (4), we will look at the minimizer $\boldsymbol{\beta}^* = Q\boldsymbol{\alpha}^*$ of the loss function (2), and in the spirit of [18], establish a bound on the difference $|\beta^*_{i1} - \beta^*_{i2}|$ of the coefficients on two similar components.

To proceed, let $\boldsymbol{\alpha}^*$ minimize (3) with: $\boldsymbol{\beta}^* = Q\boldsymbol{\alpha}^*$, $\boldsymbol{\gamma}^* = -2\lambda K \boldsymbol{\beta}^*$, and the corresponding training instance weight $\boldsymbol{w}^*$. Let $\mathcal{H}_k$ denote the subset of base classifiers acting on component $k$, i.e., $\mathcal{H}_k = \{h_j \in \mathcal{H} : s(j) = k\}$. The following lemma is proved in the supplementary material:

**Lemma:** For any $k$, $1 \leq k \leq n$, $-\gamma^*_k \geq \max_{h_j \in \mathcal{H}_k} \sum_{i=1}^m y_i h_j(x_i) w^*_i$ with equality if $\beta^*_k > 0$.

Assuming $K = \mu I - G$, $G$ defined in (4), we have the following result:

**Theorem 2.** *Let $\bar{\boldsymbol{\beta}}^* = G\boldsymbol{\beta}^*$ be the smoothed version of vector $\boldsymbol{\beta}^*$. Then for any $k_1$ and $k_2$:*

$$|\beta^*_{k1} - \beta^*_{k2}| \leq \frac{1}{\mu}|\bar{\beta}^*_{k1} - \bar{\beta}^*_{k2}| + \frac{1}{\lambda\mu}d(k_1, k_2), \tag{10}$$

*where $d(k_1, k_2) = |\max_{h_j \in \mathcal{H}_{k_1}} \sum_{i=1}^m y_i h_j(x_i) w^*_i - \max_{h_j \in \mathcal{H}_{k_2}} \sum_{i=1}^m y_i h_j(x_i) w^*_i|$.*

*Proof.* We prove the following three cases separately:

(1). $\beta^*_{k1}$ and $\beta^*_{k2}$ are both positive. In this case, using the lemma on $\gamma^*_{k1}$ and $\gamma^*_{k2}$ yields: $\left|(2\lambda\bar{\beta}^*_{k1} - 2\lambda\mu\beta^*_{k1}) - (2\lambda\bar{\beta}^*_{k2} - 2\lambda\mu\beta^*_{k2})\right| = |\gamma^*_{k1} - \gamma^*_{k2}| = d(v_{k1}, v_{k2})$. We can then use the triangle inequality on the LHS to obtain the result.

(2). One of $\beta^*_{k1}$ and $\beta^*_{k2}$ is zero the other is positive. WLOG assume $\beta^*_{k1} = 0$. Then $-\gamma^*_{k1} \geq \max_{h_j \in \mathcal{H}_{k1}} \sum_{i=1}^m y_i h_j(x_i) w^*_i$ and $-\gamma^*_{k2} = \max_{h_j \in \mathcal{H}_{k2}} \sum_{i=1}^m y_i h_j(x_i) w^*_i$. This gives:

$$\gamma^*_{k1} - \gamma^*_{k2} \leq \max_{h_j \in \mathcal{H}_{k2}} \sum_{i=1}^m y_i h_j(x_i) w^*_i - \max_{h_j \in \mathcal{H}_{k1}} \sum_{i=1}^m y_i h_j(x_i) w^*_i \leq d(v_{k1}, v_{k2}).$$

Substituting the definition of $\boldsymbol{\gamma}$: $\boldsymbol{\gamma} = 2\lambda G\boldsymbol{\beta} - 2\lambda\mu\boldsymbol{\beta} = 2\lambda\bar{\boldsymbol{\beta}} - 2\lambda\mu\boldsymbol{\beta}$, yields $(2\lambda\bar{\beta}^*_{k1} - 2\lambda\mu 0) - (2\lambda\bar{\beta}^*_{k2} - 2\lambda\mu\beta^*_{k2}) \leq d(v_{k1}, v_{k2})$. Therefore $2\lambda\mu\beta^*_{k2} \leq (2\lambda\bar{\beta}^*_{k2} - 2\lambda\bar{\beta}^*_{k1}) + d(v_{k1}, v_{k2})$. Using the triangle inequality on the right hand side of the previous expression yields the result.

(3) $\beta^*_{k1} = \beta^*_{k2} = 0$. In this case, the inequality is obvious. $\qquad\square$

The theorem upper bounds the difference in the importance coefficient of two components by the sum of two terms: the first, $|\bar{\beta}^*_{k1} - \bar{\beta}^*_{k2}|$, takes into account the importance weight of nearby locations. This term is small when the two locations are spatially close, or when they are in two neighborhoods that contain a similar amount of important voxels. The second term reflects the dissimilarity between two voxels. This term measures the difference in the weighted performances of a location's best base classifier. Clearly, $d(k_1, k_2) = 0$ when components $k_1$ and $k_2$ are identical under $\mathcal{H}$ over the training instances. More generally, we can sort all the training examples by the activation level on a single component. If sorting on locations $k_1$ and $k_2$ yields the same results, then $d(k_1, k_2) = 0$.

## 6   Experiments

The first experiment is gender classification using features located on 58 annotated landmark points in the IMM face data set [19] (Figure 3(a)). For each point we extract the first 3 principal components of a $15 \times 15$ window as features. We randomly choose 7 males and 7 females to do leave-one-out 7-fold cross-validation for 100 trials. AdaBoost yields an average classification accuracy of $\tau = 78.8\%$

with a standard deviation of $\sigma = 19.9\%$. SRB ($\lambda = 0.1$, $r = 10$ pixel-length) achieves $\tau = 80.5\%$ and $\sigma = 18.7\%$. The component importance map $\boldsymbol{\beta}$ of SRB reveals both eyes as discriminating areas and demonstrates the grouping effect. (All experiments in this section use $\mu = \max_j(\sum_i G_{ij})$. By (10), a larger $\mu$ will make this grouping effect more dominant). The $\boldsymbol{\beta}$ for AdaBoost is less smooth and less interpretable with the most important component on the left chin (Figure 3(b,c)).

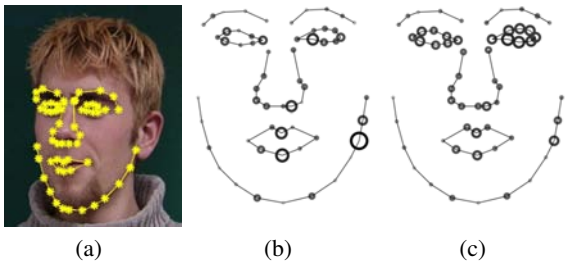

(a)      (b)      (c)

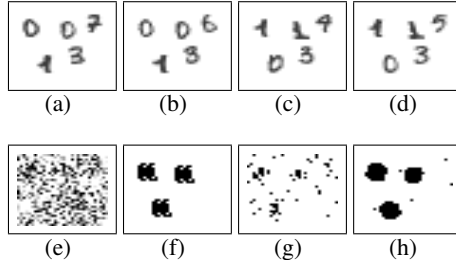

(a)      (b)      (c)      (d)

(e)      (f)      (g)      (h)

*Figure 3:* Experiment 1. (a): an example showing annotated points; (b-c): the average component importance map $\boldsymbol{\beta}$ (indicated by sizes of the circles) after running (b) AdaBoost and (c) SRB for 50 iterations.

*Figure 4:* Experiment 2. (a-d): example images; (e): example training image with noise; (f): ground truth of discriminative pixels; (g-h): pixels selected by (g) AdaBoost and (h) SRB.

The second experiment is a binary image classification task. Each image contains the handwritten digits 1,1,0,3 and a random digit, all in fixed locations. Digits 0 and 1 are swapped between the classes (Figure 4(a-d)). The handwritten digit images are from the OCR digits data set [20]. To obtain the training/testing instances we add noise to the images (Figure 4(e)). We test the ability of several algorithms to: (a) find the discriminating pixels, and (b) if a classification algorithm, accurately classify the classes. The quality of pixel selection is measured by a precision-recall curve, with ground truth pixels (Figure 4(f)) selected by a t-test on the two classes of noiseless images. This curve is plotted for the following methods: (1) SRB ($\lambda = 0.5$, $r = \frac{1}{\sqrt{2}}$ pixel-length) (2) AdaBoost; (3) thresholding the univariate t-test score; (4) thresholding the first one or two principle component(s); (5) thresholding the pixel coefficients in an LDA model with diagonal covariance (Gaussian naive bayes classifier); (6) level-set method [6] on a Z-statistics map. We plot the precision-recall curve by varying the number of iterations (for (1),(2)) or the value of the threshold (for (3)-(6)). We also tried all methods with Gaussian spatial pre-smoothing as a preprocessing step. The classification accuracies are measured for methods (1), (2) and (5) on separate test data.

The results, averaged over 100 noise realizations, are plotted in Figure 5. SRB showed no loss of classification accuracy nor convergence speed (usually within 100 iterations), and achieved the best pixel selection among all methods. It is better than Gaussian naive Bayes and PCA methods, even when the noise matches the i.i.d. Gaussian assumption of these methods (Figure 5(a,d)). In all cases, local spatial averaging deteriorates the classification performance of boosting.

In the third experiment, subjects watch a movie during the fMRI scan. The classification task is to discriminate two types of scenes (faces and objects) based on the fMRI responses. Each fMRI responses is a single TR scan of the brain volume. We divide the data (14 subjects, 26 face and 18 object fMRI responses) into 10 cross validation groups and average the classification accuracies. SRB ($\lambda = 0.1$, $r = 5$ voxel-length) trained for 100 iterations yields accuracy $\tau = 73.3\%$ with $\sigma = 9.3\%$ across 14 subjects. AdaBoost yields $\tau = 75.5\%$ with $\sigma = 4.9\%$. To make sure this is significant, we repeated the training with shuffled labels. After shuffling, $\tau = 49.7\%$, with $\sigma = 4.6\%$, which is effectively chance. We note that spatially regularized boosting yields a more clustered and interpretable selection of voxels. The result for one subject (Figure 6) shows that standard boosting (AdaBoost) selects voxels scattered in the brain, while SRB selects clustered voxels and nicely highlights the relevant FFA area [21] and posterior central sulcus [22, 23].

## 7 Conclusions

The proposed SRB algorithm is applicable to a variety of situations in which one needs to boost the performance of base classifiers with spatial structure. The mechanism of the algorithm has a

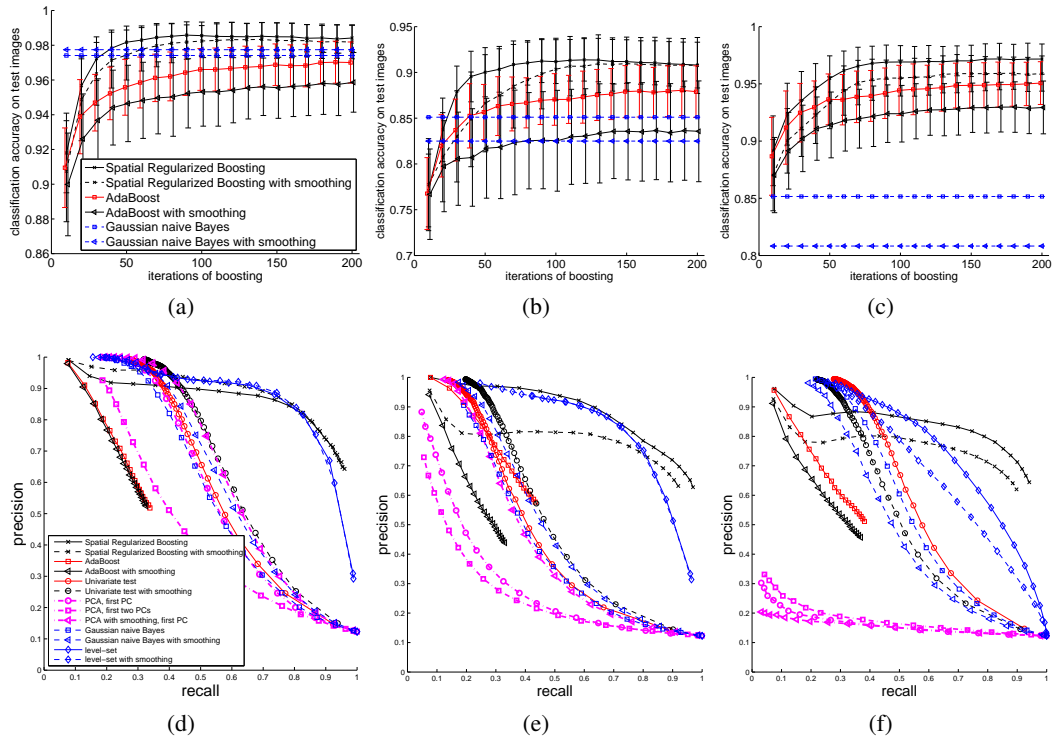

*Figure 5:* Experiment 2. (a-c): test classification accuracy: (a) i.i.d. Gaussian noise, (b) poisson noise, (c) spatially correlated Gaussian noise. (b,c) share the legend of (a). (d-f): pixel selection performances: (d) i.i.d. Gaussian noise, (e) poisson noise, (f) spatial correlated Gaussian noise. (e,f) share the legend of (d).

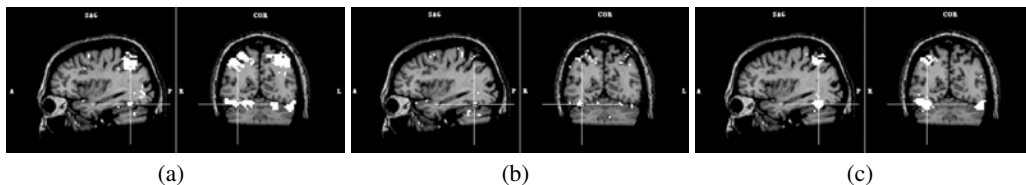

*Figure 6:* Experiment 3: an example: sets of voxels selected by (a) univariate t-test (b) AdaBoost and (c) SRB

natural interpretation: in each iteration, the algorithm selects a base classifier with the best performance evaluated under two sets of weights: weights on training examples (as in AdaBoost) and weights on locations. The additional set of location weights encourages or discourages the selection of certain base classifiers based on the spatial location of base classifiers that have already been selected. Computationally, SRB is as effective as AdaBoost. We demonstrated the effectiveness of the algorithm both by providing a theoretical analysis of the "grouping effect" and by experiments on three data sets. The grouping effect is clearly demonstrated in the face gender detection experiment. In the OCR classification experiment, the algorithm shows superior performance in pixel selection accuracy without loss of classification accuracy. The algorithm matches the performance of the state-of-the-art set estimation methods [6] that use a more complex spatial regularization and cycle spinning technique. In the fMRI experiment, the algorithm yields a clustered selection of voxels in positions relevant to the task. An alternative approach, being explored, is to combine searchlight [9] with a strong learning algorithm (e.g. SVM) to integrate spatial locality and accurate classification.

# 8   Acknowledgments

The authors thank Princeton University's J. Insley Blair Pyne Fund for seed research funding.

# References

[1] K.J. Friston, J. Ashburner, J. Heather, et al. Statistical parametric mapping. *Neuroscience Databases: A Practical Guide*, page 237, 2003.

[2] R. Heller, D. Stanley, D. Yekutieli, N. Rubin, and Y. Benjamini. Cluster-based analysis of FMRI data. *NeuroImage*, 33(2):599–608, 2006.

[3] D. Van De Ville, T. Blu, and M. Unser. Integrated wavelet processing and spatial statistical testing of fMRI data. *NeuroImage*, 23(4):1472–1485, 2004.

[4] D. Van De Ville, M.L. Seghier, F. Lazeyras, T. Blu, and M. Unser. WSPM: Wavelet-based statistical parametric mapping. *NeuroImage*, 37(4):1205–1217, 2007.

[5] Z. Harmany, R. Willett, A. Singh, and R. Nowak. Controlling the error in fmri: Hypothesis testing or set estimation? In *Biomedical Imaging, 5th IEEE International Symposium on*, pages 552–555, 2008.

[6] R.M. Willett and R.D. Nowak. Minimax optimal level-set estimation. *IEEE Transactions on Image Processing*, 16(12):2965–2979, 2007.

[7] J.V. Haxby, M.I. Gobbini, M.L. Furey, A. Ishai, J.L. Schouten, and P. Pietrini. Distributed and overlapping representations of faces and objects in ventral temporal cortex. *Science*, 293(5539):2425–2430, 2001.

[8] K.A. Norman, S.M. Polyn, G.J. Detre, and J.V. Haxby. Beyond mind-reading: multi-voxel pattern analysis of fMRI data. *Trends in Cognitive Sciences*, 10(9):424–430, 2006.

[9] N. Kriegeskorte, R. Goebel, and P. Bandettini. Information-based functional brain mapping. *Proceedings of the National Academy of Sciences*, 103(10):3863–3868, 2006.

[10] V. Koltchinskii, M. Martınez-Ramon, and S. Posse. Optimal aggregation of classifiers and boosting maps in functional magnetic resonance imaging. *Advances in Neural Information Processing Systems*, 17:705–712, 2005.

[11] M. Martínez-Ramón, V. Koltchinskii, G.L. Heileman, and S. Posse. fMRI pattern classification using neuroanatomically constrained boosting. *NeuroImage*, 31(3):1129–1141, 2006.

[12] Melissa K. Carroll, Kenneth A. Norman, James V. Haxby, and Robert E. Schapire. Exploiting spatial information to improve fmri pattern classification. In *12th Annual Meeting of the Organization for Human Brain Mapping, Florence, Italy*, 2006.

[13] J.H. Friedman. Greedy function approximation: A gradient boosting machine. *Annals of Statistics*, 29(5):1189–1232, 2001.

[14] Y. Freund and R. E. Schapire. A decision-theoretic generalization of on-line learning and an application to boosting. In *European Conference on Computational Learning Theory*, pages 23–37, 1995.

[15] C. Rudin, I. Daubechies, and R.E. Schapire. The dynamics of adaboost: Cyclic behavior and convergence of margins. *Journal of Machine Learning Research*, 5(2):1557, 2005.

[16] Z.J. Xiang and P.J. Ramadge. Sparse boosting. In *IEEE International Conference on Acoustics, Speech and Signal Processing*, 2009.

[17] T. Zhang. Adaptive Forward-Backward Greedy Algorithm for Sparse Learning with Linear Models. In *Proc. Neural Information Processing Systems*, 2008.

[18] H. Zou and T. Hastie. Regression shrinkage and selection via the elastic net, with applications to microarrays. *JR Statist. Soc. B*, 2004.

[19] M.M. Nordstrøm, M. Larsen, J. Sierakowski, and M.B. Stegmann. The IMM face database-an annotated dataset of 240 face images. Technical report, DTU Informatics, Building 321, 2004.

[20] A. Asuncion and D.J. Newman. UCI machine learning repository, 2007.

[21] N. Kanwisher, J. McDermott, and M.M. Chun. The fusiform face area: a module in human extrastriate cortex specialized for face perception. *Journal of Neuroscience*, 17(11):4302–4311, 1997.

[22] U. Hasson, M. Harel, I. Levy, and R. Malach. Large-scale mirror-symmetry organization of human occipito-temporal object areas. *Neuron*, 37(6):1027–1041, 2003.

[23] U. Hasson, Y. Nir, I. Levy, G. Fuhrmann, and R. Malach. Intersubject synchronization of cortical activity during natural vision. *Science*, 303(5664):1634–1640, 2004.

